# Context as Filtering

**Daichi Mochihashi**
ATR, Spoken Language Communication
Research Laboratories
Hikaridai 2-2-2, Keihanna Science City
Kyoto, Japan
daichi.mochihashi@atr.jp

**Yuji Matsumoto**
Graduate School of Information Science
Nara Institute of Science and Technology
Takayama 8916-5, Ikoma City
Nara, Japan
matsu@is.naist.jp

## Abstract

Long-distance language modeling is important not only in speech recognition and machine translation, but also in high-dimensional discrete sequence modeling in general. However, the problem of context length has almost been neglected so far and a naïve bag-of-words history has been employed in natural language processing. In contrast, in this paper we view topic shifts within a text as a latent stochastic process to give an explicit probabilistic generative model that has partial exchangeability. We propose an online inference algorithm using particle filters to recognize topic shifts to employ the most appropriate length of context automatically. Experiments on the BNC corpus showed consistent improvement over previous methods involving no chronological order.

## 1 Introduction

Contextual effect plays an essential role in the linguistic behavior of humans. We infer the context in which we are involved to make an adaptive linguistic response by selecting an appropriate model from that information. In natural language processing research, such models are called long-distance language models that incorporate distant effects of previous words over the short-term dependencies between a few words, which are called $n$-gram models. Besides apparent application in speech recognition and machine translation, we note that many problems of discrete data processing reduce to language modeling, such as information retrieval [1], Web navigation [2], human-machine interaction or collaborative filtering and recommendation [3].

From the viewpoint of signal processing or control theory, context modeling is clearly a filtering problem that estimates the states of a system sequentially along time to predict the outputs according to them. However, for the problem of long-distance language modeling, natural language processing has so far only provided simple averaging using a set of whole words from the beginning of a text, totally dropping chronological order and implicitly assuming that the text comes from a stationary information source [4, 5].

The inherent difficulties that have prevented filtering approaches to language modeling are its discreteness and high dimensionality, which precludes Kalman filters and their extensions that are all designed for vector spaces and distributions like Gaussians. As we note in the following, ordinary discrete HMMs are not powerful enough for this purpose because their true state is restricted to a single hidden component [6].

In contrast, this paper proposes to solve the *high-dimensional discrete filtering problem* directly using a Particle Filter. By combining a multinomial Particle Filter recently proposed in statistics for DNA sequence modeling [7] with Bayesian text models LDA and DM, we introduce two models that can track multinomial stochastic processes of natural language or similar high-dimensional discrete data domains that we often encounter.

## 2 Mean Shift Model of Context

### 2.1 HMM for Multinomial Distributions

The long-distance language models mentioned in Section 1 assume a hidden multinomial distribution, such as a unigram distribution or a mixture distribution over the latent topics, to predict the next word by updating its estimate according to the observations. Therefore, to track context shifts, we need a model that describes changes of multinomial distributions.

One model for this purpose is a multinomial extension to the Mean shift model (MSM) recently proposed in the field of statistics [7]. This is a kind of HMM, but note that it is different from traditional discrete HMMs. In discrete HMMs, the true state is one of $M$ components and we estimate it stochastically as a multinomial over the $M$ components. On the other hand, since the true state here is itself a multinomial over the components, we estimate it stochastically as (possibly a mixture of) a Dirichlet distribution, a distribution of multinomial distributions on the $(M-1)$-simplex. This HMM has some similarity to the Factorial HMM [6] in that it has a combinatorial representational power through a distributed state representation. However, because the true state here is a multinomial over the latent variables, there are dependencies between the states that are assumed independent in the FHMM. Below, we briefly introduce a multinomial Mean shift model following [7] and an associated solution using a Particle Filter.

### 2.2 Multinomial Mean Shift Model

The MSM is a generative model that describes the intermittent changes of hidden states and outputs according to them. Although there is a corresponding counterpart using Normal distribution that was first introduced [8, 9], here we concentrate on a multinomial extension of MSM, following [7] for DNA sequence modeling.

In a multinomial MSM, we assume time-dependent true multinomials $\boldsymbol{\theta}_t$ that may change occasionally and the following generative model for the discrete outputs $\mathbf{y}_t = y_1 y_2 \ldots y_t$ ($y_t \in \Sigma$ ; $\Sigma$ is a set of symbols) according to $\boldsymbol{\theta}_1 \boldsymbol{\theta}_2 \ldots \boldsymbol{\theta}_t$:

$$\begin{cases} \boldsymbol{\theta_t} \sim \mathrm{Dir}(\boldsymbol{\alpha}) & \text{with probability } \rho \\ \quad = \boldsymbol{\theta}_{t-1} & \text{with probability } (1-\rho) \,, \\ y_t \sim \mathrm{Mult}(\boldsymbol{\theta}_t) \end{cases} \tag{1}$$

where $\mathrm{Dir}(\boldsymbol{\alpha})$ and $\mathrm{Mult}(\boldsymbol{\theta})$ are a Dirichlet and multinomial distribution with parameters $\boldsymbol{\alpha}$ and $\boldsymbol{\theta}$, respectively. Here we assume that the hyperparameter $\boldsymbol{\alpha}$ is known and fixed, an assumption we will relax in Section 3.

This model first draws a multinomial $\boldsymbol{\theta}$ from $\mathrm{Dir}(\boldsymbol{\alpha})$ and samples output $y$ according to $\boldsymbol{\theta}$ for a certain interval. When a change point occurs with probability $\rho$, a new $\boldsymbol{\theta}$ is sampled again from $\mathrm{Dir}(\boldsymbol{\alpha})$ and subsequent $y$ is sampled from the new $\boldsymbol{\theta}$. This process continues recursively throughout which neither $\boldsymbol{\theta}_t$ nor the change points are known to us; all we know is the output sequence $\mathbf{y}_t$.

However, if we know that the change has occurred at time $c$, $y$ can be predicted exactly. Let $I_t$ be a binary variable that represents whether a change occurred at time $t$: that is, $I_t = 1$ means there was a change at $t$ ($\boldsymbol{\theta}_t \neq \boldsymbol{\theta}_{t-1}$), and $I_t = 0$ means there was no change ($\boldsymbol{\theta}_t = \boldsymbol{\theta}_{t-1}$). When the last change occurred at time $c$,

$$p(y_{t+1}\!=\!y\,|\,\mathbf{y}_t, I_c\!=\!1, I_{c+1}\!=\!\cdots\!=\!I_t\!=\!0) = \int p(y|\boldsymbol{\theta})p(\boldsymbol{\theta}|y_c\cdots y_t)d\boldsymbol{\theta} \qquad (2)$$
$$= \frac{\alpha_y + n_y}{\sum_y(\alpha_y+n_y)}, \qquad (3)$$

where $\alpha_y$ is the $y$'th element of $\boldsymbol{\alpha}$ and $n_y$ is the number of occurrences of $y$ in $y_c \cdots y_t$. Therefore, the essence of this problem lies in how to detect a change point given the data up to time $t$, a change point problem in discrete space. Actually, this problem can be solved by an efficient Particle Filter algorithm [10] shown below.

## 2.3 Multinomial Particle Filter

The prediction problem above can be solved by the efficient Particle Filter algorithm shown in Figure 1, graphically displayed in Figure 2 (excluding prior updates). The main intricacy involved is as follows. Let us denote $\mathbf{I}_t = \{I_1 \ldots I_t\}$. By Bayes' theorem,

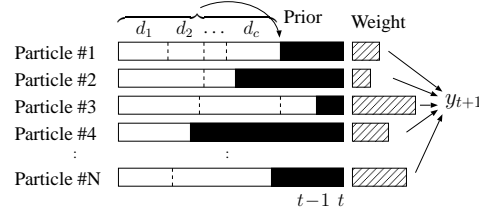

Figure 2: Multinomial Particle Filter in work.

$$p(I_t|\mathbf{I}_{t-1}, \mathbf{y}_t) \propto p(I_t, y_t|\mathbf{I}_{t-1}, \mathbf{y}_{t-1}) = p(y_t|\mathbf{y}_{t-1}, \mathbf{I}_{t-1}, I_t)p(I_t|\mathbf{I}_{t-1}) \qquad (5)$$
$$= \begin{cases} p(y_t|\mathbf{y}_{t-1}, \mathbf{I}_{t-1}, I_t\!=\!1)p(I_t\!=\!1|\mathbf{I}_{t-1}) \;=: f(t) \\ p(y_t|\mathbf{y}_{t-1}, \mathbf{I}_{t-1}, I_t\!=\!0)p(I_t\!=\!0|\mathbf{I}_{t-1}) \;=: g(t) \end{cases} \qquad (6)$$

leading

$$\begin{cases} p(I_t\!=\!1|\mathbf{I}_{t-1}, \mathbf{y}_t) = f(t)/(f(t)+g(t)) \\ p(I_t\!=\!0|\mathbf{I}_{t-1}, \mathbf{y}_t) = g(t)/(f(t)+g(t))\,. \end{cases} \qquad (7)$$

In Expression (5), the first term is a likelihood of observation $y_t$ when $\mathbf{I}_t$ has been fixed, which can be obtained through (3). The second term is a prior probability of change, which can be set tentatively by a constant $\rho$. However, when we endow $\rho$ with a prior Beta distribution $\text{Be}(\alpha, \beta)$, posterior estimate of $\rho_t$ given the binary change point history $\mathbf{I}_{t-1}$ can be obtained using the number of 1's in $\mathbf{I}_{t-1}$, $n_{t-1}(1)$, following a standard Bayesian method:

$$E[\rho_t|\mathbf{I}_{t-1}] = \frac{\alpha + n_{t-1}(1)}{\alpha + \beta + t - 1}\,. \qquad (8)$$

This means that we can estimate a "rate of topic shifts" as time proceeds in a Bayesian fashion. Throughout the following experiments, we used this online estimate of $\rho_t$.

The above algorithm runs for each observation $y_t$ $(t\!=\!1\ldots T)$. If we observe a "strange" word that is more predictable from the prior than the contextual distribution, (6) makes $f(t)$ larger than $g(t)$, which leads to a higher probability that $I_t\!=\!1$ will be sampled in the Bernoulli trial of Algorithm 1(b).

# 3 Mean Shift Model of Natural Language

Chen and Lai [7] recently proposed the above algorithm to analyze DNA sequences. However, when extending this approach to natural language, i.e. word sequences, we meet two serious problems.

The first problem is that in a natural language the number of words is extremely large. As opposed to DNA, which has only four letters of A/T/G/C, a natural language usually contains a minimum of some tens of thousands of words and there are strong correlations between them. For example, if "nurse" follows "hospital" we believe that there has been no context shift; however, if "university" follows "hospital," the context probably has been shifted to a "medical school" subtopic, even though the two words are equally distinct from "hospital." Of course, this is due to the semantic relationship we can assume between these words. However, the original multinomial MSM cannot capture this relationship because it treats the words independently. To incorporate this relationship, we require an extensive prior knowledge of words as a probabilistic model.

The second problem is that in model equation (1), the hyperparameter $\boldsymbol{\alpha}$ of prior Dirichlet distribution of the latent multinomials is assumed to be known. In the case of natural language, this means we know beforehand what words or topics will be spoken for all the texts. Apparently, this is not a natural assumption: we need an online estimation of $\boldsymbol{\alpha}$ as well when we want to extend MSM to natural languages.

To solve these problems, we extended a multinomial MSM using two probabilistic text models, LDA and DM. Below we introduce MSM-LDA and MSM-DM, in this order.

## 3.1 MSM-LDA

Latent Dirichlet Allocation (LDA) [3] is a probabilistic text model that assumes a hidden multinomial topic distribution $\boldsymbol{\theta}$ over the $M$ topics on a document $d$ to estimate it stochastically as a Dirichlet distribution $p(\boldsymbol{\theta}|d)$. Context modeling using LDA [5] regards a history $\mathbf{h} = w_1 \ldots w_h$ as a pseudo document and estimates a variational approximation $q(\boldsymbol{\theta}|\mathbf{h})$ of a topic distribution $p(\boldsymbol{\theta}|\mathbf{h})$ through a variational Bayes EM algorithm on a document [3]. After obtaining topic distribution $q(\boldsymbol{\theta}|\mathbf{h})$, we can predict the next word as follows.

$$p(y|\mathbf{h}) = \int p(y|\boldsymbol{\theta})q(\boldsymbol{\theta}|\mathbf{h})d\boldsymbol{\theta} = \sum_{i=1}^{M} p(y|\theta_i)\langle\theta_i\rangle_{q(\boldsymbol{\theta}|\mathbf{h})} \qquad (9)$$

When we use this prediction with an associated VB-EM algorithm in place of the naïve Dirichlet model (3) of MSM, we get an MSM-LDA that tracks a latent topic distribution $\boldsymbol{\theta}$ instead of a word distribution. Since each particle computes a Dirichlet posterior of topic distribution, the final topic distribution of MSM-LDA is a mixture of Dirichlet distributions for predicting the next word through (4) and (9) as shown in Figure 3(a). Note that MSM-LDA has an implicit generative model corresponding to (1) in topic space. However, here we use a conditional model where LDA parameters are already known in order to estimate the context online.

In MSM-LDA, we can also update the hyperparameter $\boldsymbol{\alpha}$ sequentially from the history. As seen in Figure 2, each particle has a history that has been segmented into pseudo "documents" $d_1 \ldots d_c$ by the change points sampled so far. Since each pseudo "document" has a Dirichlet posterior $q(\boldsymbol{\theta}|d_i)$ ($i = 1 \ldots c$), a common Dirichlet prior can be inferred by a linear-time Newton-Raphson algorithm [3]. Note that this computation needs only be run when a change point has been sampled. For this purpose, only the sufficient statistics $q(\boldsymbol{\theta}|d_i)$ must be stored for each particle to render itself an online algorithm.

Note in passing that MSM-LDA is a model that only tracks a mixing distribution of a mixture model. Therefore, in principle this model is also applicable to other mixture models, e.g. Gaussian mixtures, where mixing distribution is not static but evolves according to (1).

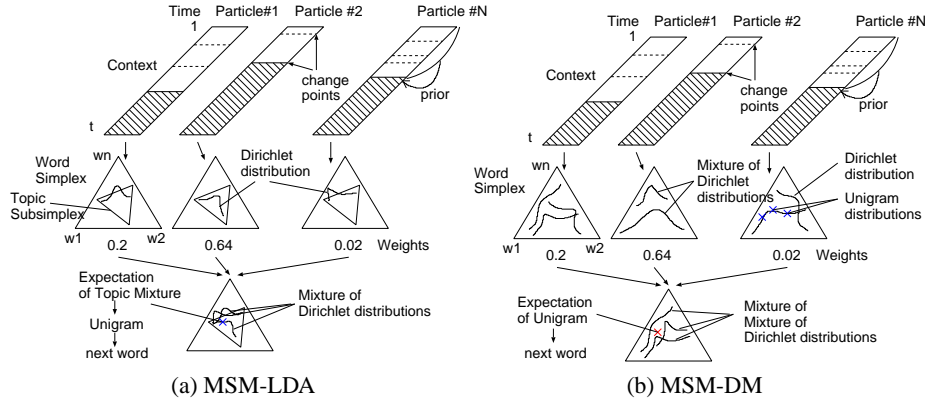

(a) MSM-LDA          (b) MSM-DM

Figure 3: MSM-LDA and MSM-DM in work.

However, in terms of multinomial estimation, this generality has a drawback because it uses a lower-dimensional topic representation to predict the next word, which may cause a loss of information. In contrast, MSM-DM is a model that works directly on the word space to predict the next word with no loss of information.

## 3.2 MSM-DM

Dirichlet Mixtures (DM) [11] is a novel Bayesian text model that has the lowest perplexity reported so far in context modeling. DM uses no intermediate "topic" variables, but places a *mixture of* Dirichlet distributions directly on the word simplex to model word correlations. Specifically, DM assumes the following generative model for a document $\mathbf{w} = w_1 \ldots w_N$:[1]

1. Draw $m \sim \text{Mult}(\boldsymbol{\lambda})$.
2. Draw $\boldsymbol{p} \sim \text{Dir}(\boldsymbol{\alpha}_m)$.
3. For $n = 1 \ldots N$,
   a. Draw $w_n \sim \text{Mult}(\boldsymbol{p})$.

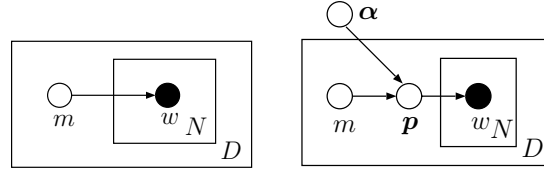

(a) Unigram Mixture (UM)    (b) Dirichlet Mixtures (DM)

Figure 4: Graphical models of UM and DM.

where $\boldsymbol{p}$ is a $V$-dimensional unigram distribution over words, $\boldsymbol{\alpha}_1 \ldots \boldsymbol{\alpha}_M = \boldsymbol{\alpha}_1^M$ are parameters of Dirichlet prior distributions of $\boldsymbol{p}$, and $\boldsymbol{\lambda}$ is a $M$-dimensional prior mixing distribution of them. This model is considered a Bayesian extension of the Unigram Mixture [12] and has a graphical model shown in Figure 4. Given a set of documents $\mathcal{D} = \{\mathbf{w}_1, \mathbf{w}_2, \ldots, \mathbf{w}_D\}$, parameters $\boldsymbol{\lambda}$ and $\boldsymbol{\alpha}_1^M$ can be iteratively estimated by a combination of EM algorithm and the modified Newton-Raphson method shown in Figure 5, which is a straight extension to the estimation of a Polya mixture [13]. [2]

Under DM, a predictive probability $p(y|\mathbf{h})$ is (omitting dependencies on $\boldsymbol{\lambda}$ and $\boldsymbol{\alpha}_1^M$):

$$p(y|\mathbf{h}) = \sum_{m=1}^{M} p(y|m, \mathbf{h})p(m|\mathbf{h}) = \sum_{m=1}^{M} \left( \int p(y|\boldsymbol{p})p(\boldsymbol{p}|\boldsymbol{\alpha}_m, \mathbf{h})d\boldsymbol{p} \right) \cdot p(m|\mathbf{h})$$
$$= \sum_{m=1}^{M} C_m \frac{\alpha_{my} + n_y}{\sum_y (\alpha_{my} + n_y)} , \tag{10}$$

**E step:** $\quad p(m|\mathbf{w}_i) \propto \lambda_m \dfrac{\Gamma(\sum_v \alpha_{mv})}{\Gamma(\sum_v \alpha_{mv} + \sum_v n_{iv})} \displaystyle\prod_{v=1}^{V} \dfrac{\Gamma(\alpha_{mv} + n_{iv})}{\Gamma(\alpha_{mv})}$ (13)

**M step:** $\quad\quad\quad \lambda_m \propto \sum_{i=1}^{D} p(m|\mathbf{w}_i),$ (14)

$$\alpha'_{mv} = \alpha_{mv} \cdot \frac{\sum_i p(m|\mathbf{w}_i)\, n_{iv}/(\alpha_{mv} + n_{iv} - 1)}{\sum_i p(m|\mathbf{w}_i) \sum_v n_{iv}/(\sum_v \alpha_{mv} + \sum_v n_{iv} - 1)} \quad (15)$$

Figure 5: EM-Newton algorithm of Dirichlet Mixtures.

where

$$C_m \propto \lambda_m \frac{\Gamma(\sum_v \alpha_{mv})}{\Gamma(\sum_v \alpha_{mv} + h)} \prod_{v=1}^{V} \frac{\Gamma(\alpha_{mv} + n_v)}{\Gamma(\alpha_{mv})} \quad (11)$$

and $n_v$ is the number of occurrences of $v$ in $\mathbf{h}$. This prediction can also be considered an extension to Dirichlet smoothing [15] with multiple hyperparameters $\boldsymbol{\alpha}_m$ to weigh them accordingly by $C_m$.[3]

When we replace a naïve Dirichlet model (3) by a DM prediction (10), we get a flexible MSM-DM dynamic model that works on word simplex directly. Since the original multinomial MSM places a Dirichlet prior in the model (1), MSM-DM is considered a natural extension to MSM by placing a mixture of Dirichlet priors rather than a single Dirichlet prior for multinomial unigram distribution. Because each particle calculates a mixture of Dirichlet posteriors for the current context, the final MSM-DM estimate is a mixture of them, again a mixture of Dirichlet distributions as shown in Figure 3(b).

In this case, we can also update the mixture prior $\boldsymbol{\lambda}$ sequentially. Because each particle has "pseudo documents" $\mathbf{w}_1 \ldots \mathbf{w}_c$ segmented by change points individually, posterior $\lambda_m$ can be obtained similarly as (14),

$$\lambda_m \propto \sum_{i=1}^{c} p(m|\mathbf{w}_i) \quad (12)$$

where $p(m|\mathbf{w}_i)$ is obtained from (13). Also in this case, only the sufficient statistics $p(m|\mathbf{w}_i)$ $(i = 1 .. c)$ must be stored to make MSM-DM a filtering algorithm.

## 4 Experiments

We conducted experiments using a standard British National Corpus (BNC). We randomly selected 100 files of BNC written texts as an evaluation set, and the remaining 2,943 files as a training set for parameter estimation of LDA and DM in advance.

### 4.1 Training and evaluation data

Since LDA and DM did not converge on the long texts like BNC, we divided training texts into pseudo documents with a minimum of ten sentences for parameter estimation. Due to the huge size of BNC, we randomly selected a maximum of 20 pseudo documents from each of the 2,943 files to produce a final corpus of 56,939 pseudo documents comprising 11,032,233 words. We used a lexicon of 52,846 words with a frequency $\geq 5$. Note that this segmentation is optional and has an only indirect influence on the experiments. It only affects the clustering of LDA and DM: in fact, we could use another corpus, e.g. newspaper corpus, to estimate the parameters without any preprocessing.

Since the proposed method is an algorithm that simultaneously captures topic shifts and their rate in a text to predict the next word, we need evaluation texts that have different rates of topic shifts. For this purpose, we prepared four different text sets by sampling

| Text | MSM-DM | DM | MSM-LDA | LDA |
|------|--------|-----|---------|-----|
| Raw | **870.06** ($-6.02\%$) | 925.83 | **1028.04** | 1037.42 |
| Slow | **893.06** ($-8.31\%$) | 974.04 | **1047.08** | 1060.56 |
| Fast | **898.34** ($-9.10\%$) | 988.26 | **1044.56** | 1061.01 |
| VFast | **960.26** ($-7.57\%$) | 1038.89 | 1065.15 | 1050.83 |

Table 2: Contextual Unigram Perplexities for Evaluation Texts.

from the long BNC texts. Specifically, we conducted sentence-based random sampling as follows.

(1) Select a first sentence randomly for each text.
(2) Sample contiguous $X$ sentences from that sentence.
(3) Skip $Y$ sentences.
(4) Continue steps (2) and (3) until a desired length of text is obtained.

In the procedure above, $X$ and $Y$ are random variables that have uniform distributions given in Table 1. We sampled 100 sentences from each of the 100 files by this procedure to create the four evaluation text sets listed in the table.

## 4.2 Parameter settings

The number of latent classes in LDA and DM are set to 200 and 50, respectively.[4] The number of particles is set to $N = 20$, a relatively small number because each particle executes an exact Bayesian prediction once previous

| Name | Property |
|------|----------|
| Raw | $X = 100, Y = 0$ |
| Slow | $1 \leq X \leq 10, 1 \leq Y \leq 3$ |
| Fast | $1 \leq X \leq 10, 1 \leq Y \leq 10$ |
| VeryFast | $X = 1, 1 \leq Y \leq 10$ |

Table 1: Types of Evaluation Texts.

change points have been sampled. Beta prior distribution of context change can be initialized as a uniform distribution, $(\alpha, \beta) = (1, 1)$. However, based on a preliminary experiment we set it to $(\alpha, \beta) = (1, 50)$: this means we initially assume a context change rate of once every 50 words in average, which will be updated adaptively.

## 4.3 Experimental results

Table 2 shows the unigram perplexity of contextual prediction for each type of evaluation set. Perplexity is a reciprocal of the geometric average of contextual predictions, thus better predictions yield lower perplexity. While MSM-LDA slightly improves LDA due to the topic space compression explained in Section 3.1, MSM-DM yields a consistently better prediction, and its performance is more significant for texts whose subtopics change faster.

Figure 6 shows a plot of the actual improvements relative to DM, $\mathrm{PPL_{MSM}} - \mathrm{PPL_{DM}}$. We can see that prediction improves for most documents by automatically selecting appropriate contexts. The maximum improvement was –365 in PPL for one of the evaluation texts. Finally, we show in Figure 7 a sequential plot of context change probabilities $p^{(i)}(I_t = 1)$ ($i = 1..N, t = 1..T$) calculated by each particle for the first 1,000 words of one of the evaluation texts.

## 5 Conclusion and Future Work

In this paper, we extended the multinomial Particle Filter of a small number of symbols to natural language with an extremely large number of symbols. By combining original filter with Bayesian text models LDA and DM, we get two models, MSM-LDA and MSM-DM, that can incorporate semantic relationship between words and can update their hyperparam-

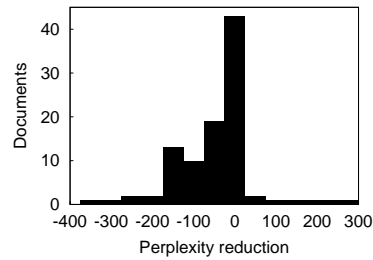

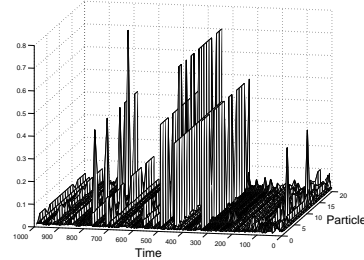

Figure 6: Perplexity reductions of MSM relative to DM.

Figure 7: Context change probabilities for 1,000 words text, sampled by the particles.

eter sequentially. According to this model, prediction is made using a mixture of different context lengths sampled by each Monte Carlo particle.

Although the proposed method is still in its fundamental stage, we are planning to extend it to larger units of change points beyond words, and to use a forward-backward MCMC or Expectation Propagation to model a semantic structure of text more precisely.

## Footnotes

[1]Step 1 of the generative model in fact can be replaced by a Dirichlet process prior. Full Bayesian treatment of DM through Dirichlet processes is now under our development.

[2]DM is an extension to the model for amino acids [14] to natural language with a huge number of parameters, which precludes the ordinary Newton-Raphson algorithm originally proposed in [14].

[3]Therefore, MSM-DM is considered an ingenious dynamic Dirichlet smoothing as well as a context modeling.

[4]We deliberately chose a smaller number of mixtures in DM because it is reported to have a better performance in small mixtures since it is essentially a unitopic model, in contrast to LDA.

## References

[1] Jay M. Ponte and W. Bruce Croft. A Language Modeling Approach to Information Retrieval. In *Proc. of SIGIR '98*, pages 275–281, 1998.

[2] David Cohn and Thomas Hofmann. The Missing Link: a probabilistic model of document content and hypertext connectivity. In *NIPS 2001*, 2001.

[3] David M. Blei, Andrew Y. Ng, and Michael I. Jordan. Latent Dirichlet Allocation. *Journal of Machine Learning Research*, 3:993–1022, 2003.

[4] Daniel Gildea and Thomas Hofmann. Topic-based Language Models Using EM. In *Proc. of EUROSPEECH '99*, pages 2167–2170, 1999.

[5] Takuya Mishina and Mikio Yamamoto. Context adaptation using variational Bayesian learning for ngram models based on probabilistic LSA. *IEICE Trans. on Inf. and Sys.*, J87-D-II(7):1409–1417, 2004.

[6] Zoubin Ghahramani and Michael I. Jordan. Factorial Hidden Markov Models. In *Advances in Neural Information Processing Systems (NIPS)*, volume 8, pages 472–478. MIT Press, 1995.

[7] Yuguo Chen and Tze Leung Lai. Sequential Monte Carlo Methods for Filtering and Smoothing in Hidden Markov Models. Discussion Paper 03-19, Institute of Statistics and Decision Sciences, Duke University, 2003.

[8] H. Chernoff and S. Zacks. Estimating the Current Mean of a Normal Distribution Which is Subject to Changes in Time. *Annals of Mathematical Statistics*, 35:999–1018, 1964.

[9] Yi-Chin Yao. Estimation of a noisy discrete-time step function: Bayes and empirical Bayes approaches. *Annals of Statistics*, 12:1434–1447, 1984.

[10] Arnaud Doucet, Nando de Freitas, and Neil Gordon. *Sequential Monte Carlo Methods in Practice*. Statistics for Engineering and Information Science. Springer-Verlag, 2001.

[11] Mikio Yamamoto and Kugatsu Sadamitsu. Dirichlet Mixtures in Text Modeling. CS Technical Report CS-TR-05-1, University of Tsukuba, 2005. http://www.mibel.cs.tsukuba.ac.jp/~myama/pdf/dm.pdf.

[12] Kamal Nigam, Andrew K. McCallum, Sebastian Thrun, and Tom M. Mitchell. Text Classification from Labeled and Unlabeled Documents using EM. *Machine Learning*, 39(2/3):103–134, 2000.

[13] Thomas P. Minka. Estimating a Dirichlet distribution, 2000. http://research.microsoft.com/~minka/papers/dirichlet/.

[14] K. Sjölander, K. Karplus, M.P. Brown, R. Hughey, R. Krogh, I.S. Mian, and D. Haussler. Dirichlet Mixtures: A Method for Improved Detection of Weak but Significant Protein Sequence Homology. *Computing Applications in the Biosciences*, 12(4):327–245, 1996.

[15] D. J. C. MacKay and L. Peto. A Hierarchical Dirichlet Language Model. *Natural Language Engineering*, 1(3):1–19, 1994.
